# A Reconfigurable Analog VLSI Neural Network Chip

**Srinagesh Satyanarayana and Yannis Tsividis**
Department of Electrical Engineering
and
Center for Telecommunications Research
Columbia University, New York, NY 10027, USA

**Hans Peter Graf**
AT&T
Bell Laboratories
Holmdel, NJ 07733
USA

## ABSTRACT

1024 distributed-neuron synapses have been integrated in an active area of 6.1mm × 3.3mm using a $0.9\mu$m, double-metal, single-poly, n-well CMOS technology. The distributed-neuron synapses are arranged in blocks of 16, which we call '4 × 4 tiles'. Switch matrices are interleaved between each of these tiles to provide programmability of interconnections. With a small area overhead (15 %), the 1024 units of the network can be rearranged in various configurations. Some of the possible configurations are, a 12-32-12 network, a 16-12-12-16 network, two 12-32 networks etc. (the numbers separated by dashes indicate the number of units per layer, including the input layer). Weights are stored in analog form on MOS capacitors. The synaptic weights are usable to a resolution of 1% of their full scale value. The limitation arises due to charge injection from the access switch and charge leakage. Other parameters like gain and shape of nonlinearity are also programmable.

## Introduction

A wide variety of problems can be solved by using the neural network framework [1]. However each of these problems requires a different topology and weight set. At a much lower system level, the performance of the network can be improved by selecting suitable neuron gains and saturation levels. Hardware realizations of

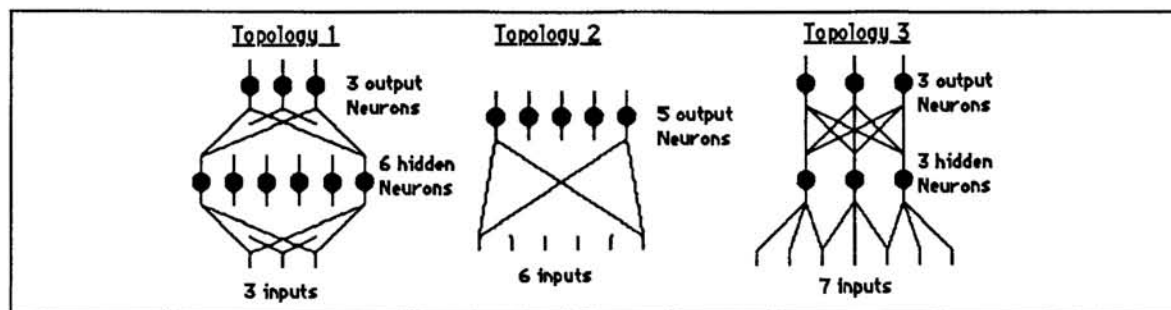

**Figure 1:** Reconfigurability

neural networks provide a fast means of solving the problem. We have chosen analog circuits to implement neural networks because they provide high synapse density and high computational speed. In order to provide a general purpose hardware for solving a wide variety of problems that can be mapped into the neural network framework, it is necessary to make the topology, weights and other neurosynaptic parameters programmable. Weight programmability has been extensively dealt in several implementations [2 - 9]. However features like programmable topology, neuron gains and saturation levels have not been addressed extensively. We have designed, fabricated and tested an analog VLSI neural network in which the topology, weights and neuron gains and saturations levels are all programmable.

Since the process of design, fabrication and testing is time-consuming and expensive, redesigning the hardware for each application is inefficient. Since the field of neural networks is still in its infancy, new solutions to problems are being searched for everyday. These involve modifying the topology [10] and finding the best weight set. In such an environment, a computational tool that is fully programmable is very desirable.

## The Concept of Reconfigurability

We define *reconfigurability* as the ability to alter the topology (the number of layers, number of neurons per layer , interconnections from layer to layer and interconnections within a layer) of the network. The topology of a network does not describe the value of each synaptic weight. It only specifies the presence or absence of a synapse between two neurons (However in the special case of binary weight (0,1), defining the topology specifies the weight). The ability to alter the synaptic weight can be defined as weight programmability. Figure 1 illustrates reconfigurability, whereas Figure 2 shows how the weight value is realized in our implementation. The Voltage $V_w$ across the capacitor represents the synaptic weight. Altering this voltage makes weight programmability possible.

## Why is On-Chip Reconfigurability Important ?

Synapses, neurons and interconnections occupy real estate on a chip. Chip sizes are limited due to various factors like yield and cost. Hence only a limited number

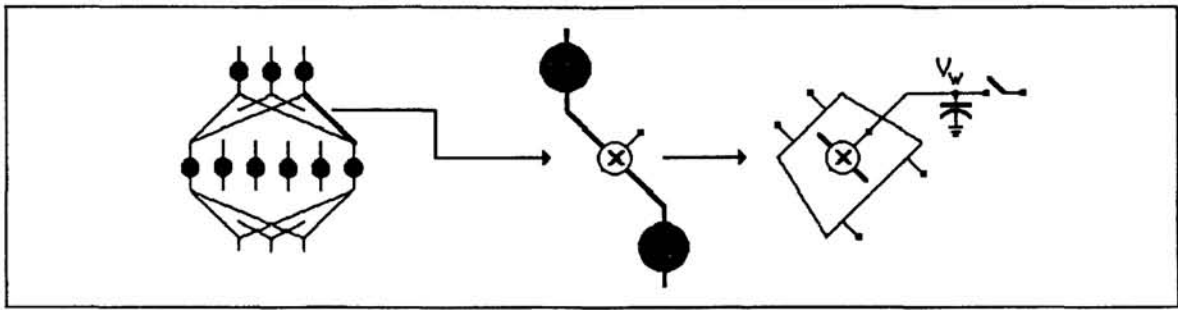

**Figure 2:** Weight programmability

of synapses can be integrated in a given chip area. Currently the most compact realizations (considering more than 6 bits of synaptic accuracy) permit us to integrate only a few thousand synapses per $cm^2$. In such a situation every zero-valued (inactive) synapse represents wasted area, and decreases the computational ability per unit area of the chip. If a fixed topology network is used for different problems, it will be underutilized as long as some synapses are set to zero value. On the other hand, if the network is reconfigurable, the limited resources on-chip can be reallocated to build networks with different topologies more efficiently. For example the network with topology-2 of Figure 1 requires 30 synapses. If the network was reconfigurable, we could utilize these synapses to build a two-layer network with 15 synapses in the first layer and 15 in the second layer. In a similar fashion we could also build the network with topology-3 which is a network with localized receptive fields.

## The Distributed-Neuron Concept

In order to provide reconfigurability on-chip, we have developed a new cell called the distributed-neuron synapse [11]. In addition to making reconfiguration easy, it has other advantages like being modular hence making design easy, provides automatic gain scaling , avoids large current build-up at any point and makes possible a fault tolerant system.

Figure 3 shows a lumped neuron with $N$ synaptic inputs. We call it 'lumped' because, the circuit that provides the nonlinear function is lumped into one block.

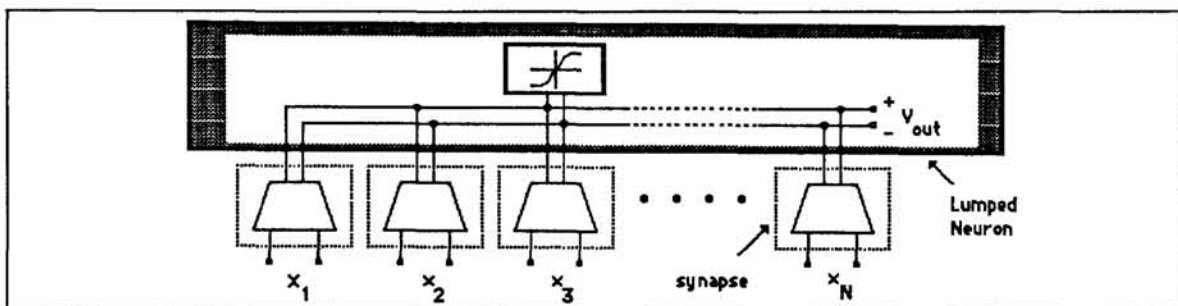

**Figure 3:** A lumped neuron with $N$ synaptic inputs

The synapses are assumed to be voltage-to-current (transconductor) cells, and the neuron is assumed to be a current-to-voltage cell. Summation is achieved through addition of the synapse output currents in the parallel connection.

Figure 4 shows the equivalent distributed-neuron with $N$ synaptic inputs. It is called 'distributed' because the circuit that functions as the neuron, is split into 'N' parts. One of these parts is integrated with each synapse. This new block ( that contains a a synapse and a fraction of the neuron ) is called the 'distributed-neuron synapse'. Details of the distributed-neuron concept are described in [11]. It has to be noted that the splitting of the neuron to form the distributed-neuron synapse is done at the summation point where the computation is linear. Hence the two realizations of the neuron are computationally equivalent. However, the distributed-neuron implementation offers a number of advantages, as is now explained.

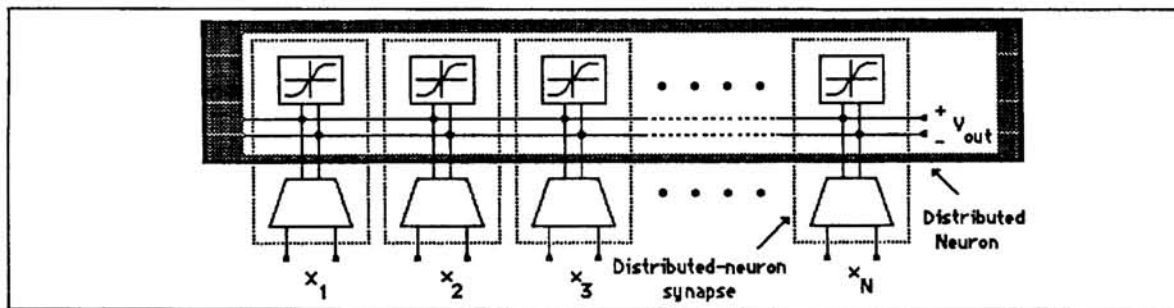

**Figure 4:** A distributed-neuron with $N$ synaptic inputs

## Modularity of the design

As is obvious from Figure 4, the task of building a complete network involves designing one single distributed-neuron synapse module and interconnecting several of them to form the whole system. Though at a circuit level, a fraction of the neuron has to be integrated with each synapse, the system level design is simplified due to the modularity.

## Automatic gain normalization

In the distributed-neuron, each unit of the neuron serves as a load to the output of a synapse. As the number of synapses at the input of a neuron increases, the number of neuron elements also increases by the same number. The neuron output is given by:

$$y_j = f\{\frac{1}{N}\sum_{i=1}^{N} w_{ij}x_i - \Theta j\} \tag{1}$$

Where $y_j$ is the output of the $j^{th}$ neuron, $w_{ij}$ is the weight from the $i^{th}$ synaptic input $x_i$ and $\Theta_j$ is the threshold, implemented by connecting in parallel an appropriate number of distributed-neuron synapses with fixed inputs. Assume for the

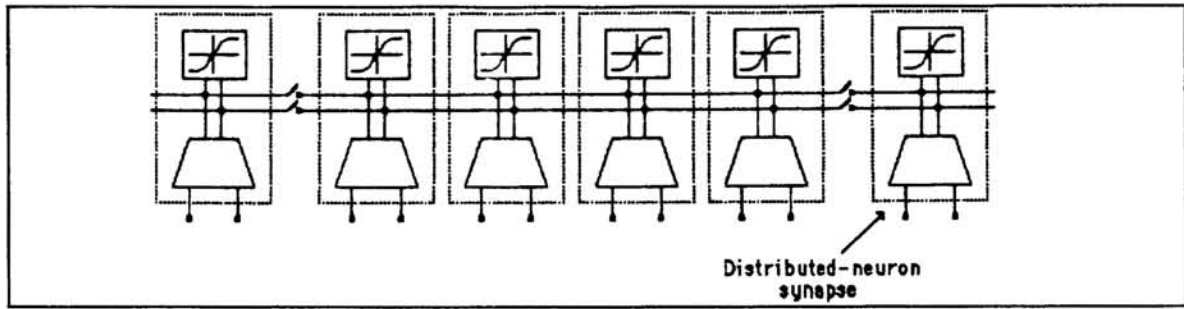

Distributed-neuron
synapse

**Figure 5:** Switches used for reconfiguration in the distributed-neuron implementation.

moment that all the inputs $x_i$ are at a maximum possible value. Then it is easily seen that $y_j$ is independent of $N$ . This is the manifestation of the automatic gain normalization that is inherent to the idea of distributed-neuron synapses.

### Ease of reconfiguration

In a distributed-neuron implementation, reconfiguration involves interconnecting a set of distributed-neuron synapse modules (Figure 5). A neuron of the right size gets formed when the outputs of the required number of synapses are connected. In a lumped neuron implementation, reconfiguration involves interconnecting a set of synapses with a set of neurons. This involves more wiring, switches and logic control blocks.

### Avoiding large current build-up in the neuron

In our implementation the synaptic outputs are currents. These currents are summed by Kirchoffs current law and sent to the neuron. Since the neuron is distributed, the total current is divided into $N$ equal parts, where $N$ is the number of distributed-neuron synapses. One of these part flows through each unit of the distributed neuron as illustrated in Figure 4. This obviates the need for large current summation wires and avoids other problems associated with large currents at any single point.

### Fault tolerance

On a VLSI chip defects are commonly seen. Some of these defects can short wires, hence corrupting the signals that are carried on them. Defects can also render some synapses and neurons defective. In our implementation, we have integrated switches in-between groups of distributed-neuron synapses (which we call 'tiles') to make the chip reconfigurable (Figure 6). This makes each tile of the chip externally testable. The defective sections of the chip can be isolated and the remaining synapses can thus be reconfigured into another topology as shown in Figure 6.

## Circuit Description of the Distributed-Neuron Synapse

Figure 7 shows a distributed-neuron synapse constructed around a differential-input, differential-output transconductance multiplier. A weight converter is used to con-

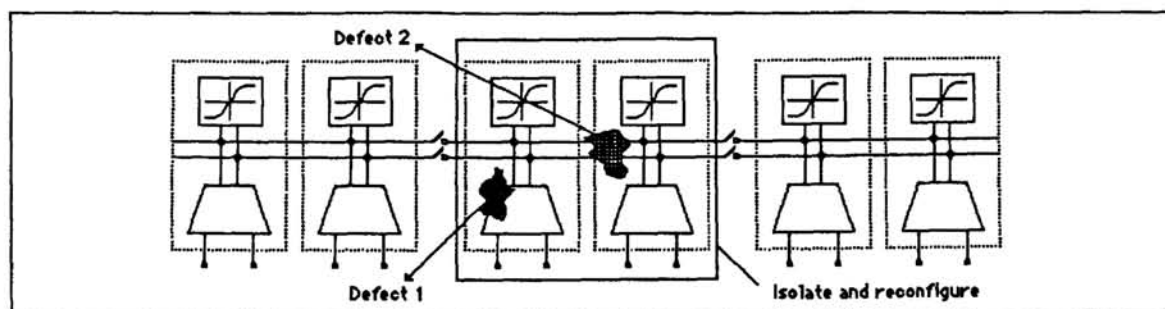

**Figure 6:** Improved fault tolerance in the distributed-neuron system

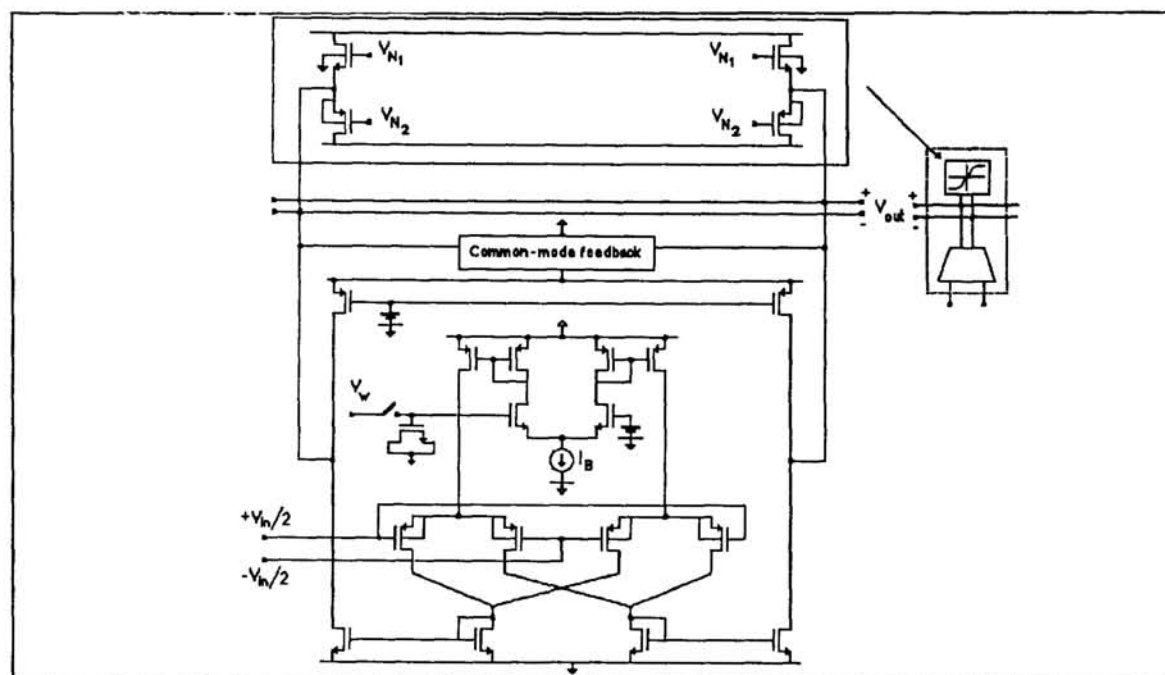

**Figure 7:** The distributed-neuron synapse circuit

vert the single-ended weight controlling voltage $V_w$ into a set of differential currents that serve as the bias currents of the multiplier. The weight is stored on a MOS capacitor.

The differential nature of the circuit offers several advantages like improved rejection of power supply noise and linearity of multiplication. Common-mode feedback is provided at the output of the synapse. An amplitude limiter that is operational only when the weighted sum exceeds a certain range serves as the distributed-neuron part. The saturation levels of the neuron can be programmed by adjusting $V_{N_1}$ and $V_{N_2}$. Gains can be set by adjusting the bias current $I_B$ and/or a load (not shown). The measured synapse characteristics are shown in Figure 8 .

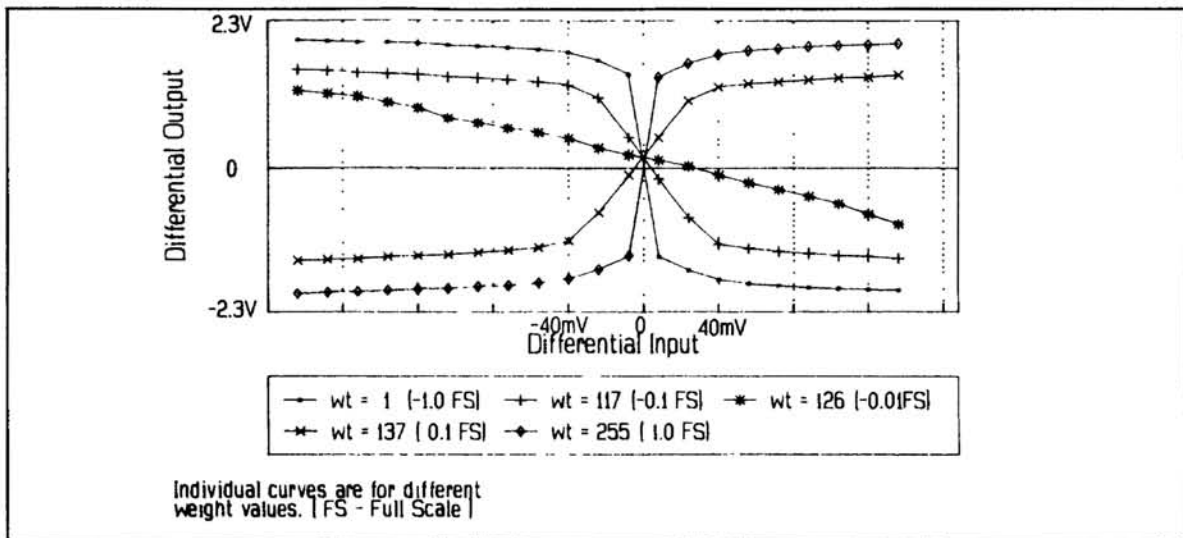

**Figure 8:** Measured characteristics of the distributed-neuron synapse

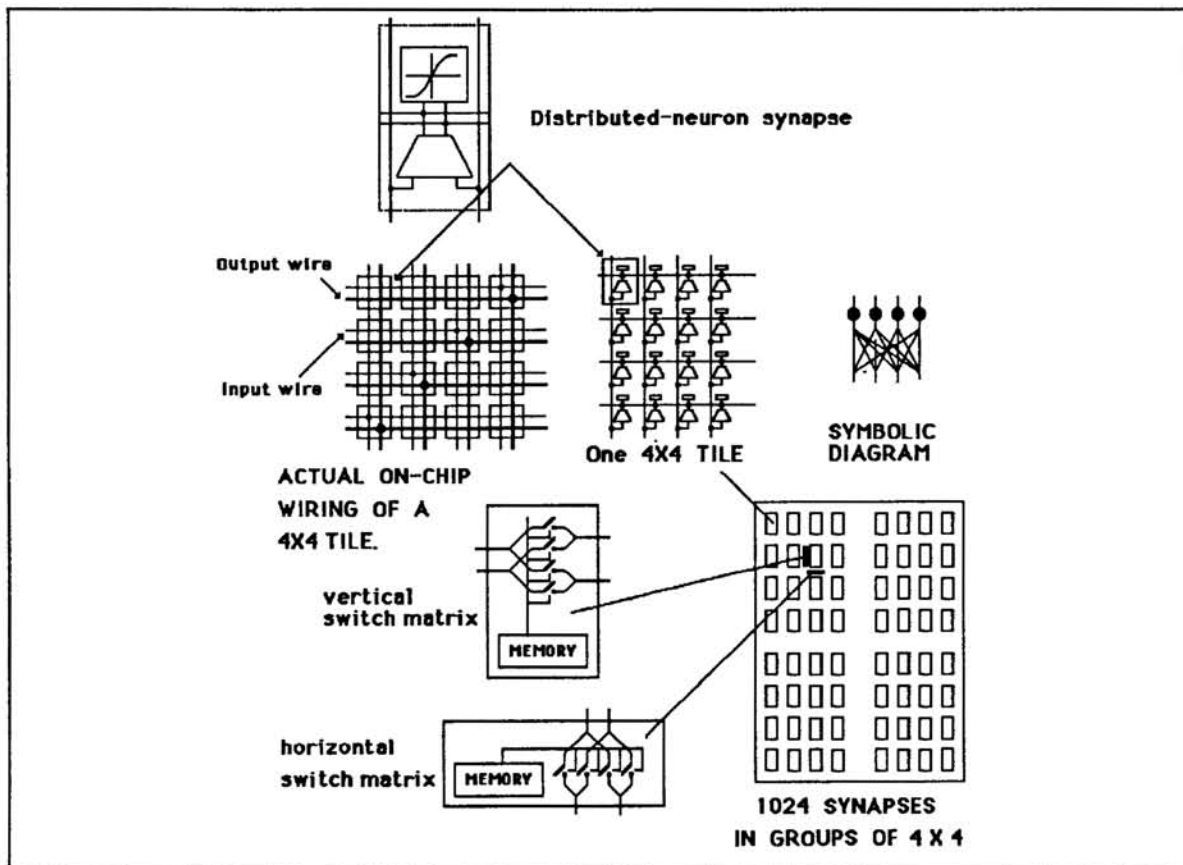

**Figure 9:** Organization of the distributed-neurons and switches on chip

## Organization of the Chip

Figure 9 shows how the distributed-neuron synapses are arranged on-chip. 16 distributed-neuron synapses have been arranged in a 4 × 4 crossbar fashion to form a 4-input-4-output network. We call this a '4 × 4 tile'. Input and output wires are available on all four sides of the tile. This makes interconnections to adjacent blocks easy. Vertical and horizontal switch matrices are interleaved in-between the tiles to select one of the various possible modes of interconnections. These modes can be configured by setting the 4 bits of memory in each switch matrix. 1024 distributed-neuron synapses have been integrated in an active area of 6.1mm × 3.3mm using a 0.9$\mu$m, double-metal, single-poly, n-well CMOS technology.

## The Weight Update/Refresh Scheme

Weights are stored in analog form on a MOS capacitor. A semi-serial-parallel weight update scheme has been built. 8 pins of the chip are used to distribute the weights to the 1024 capacitors on the chip. Each pin can refresh 128 capacitors contained in a row of tiles. The capacitors in each tile-row are selected one at a time by a decoder. The maximum refresh speed depends on the time needed to charge up the weight storage capacitor and the parasitic capacitances. One complete refresh of all weights on the chip is possible in about 130 $\mu$ seconds. However one could refresh at a much slower rate, the lower limit of which is decided by the charge leakage. For a 7-bit precision in the weight at room temperature, a refresh rate in the order of milliseconds should be adequate. Charge injection due to the parasitic capacitances has been kept low by using very small switches. In the first version of the chip, only the distributed-neuron synapses, the switches used for reconfiguration, and the topology memory have been integrated. Weights are stored outside the chip in digital form in a 1K × 8 RAM. The contents of the RAM are continuously read and converted into analog form using a bank of off-chip D/A converters. An advantage of our scheme is that the forward-pass operation is not interrupted by the weight refresh mechanism. A fast weight update scheme of the type used here is very desirable while executing learning algorithms at a high speed. The complete block diagram of the weight refresh/update and testing scheme is shown in Figure 10.

## Configuration Examples

In Figure 11 we show some of the network topologies that can be configured with the resources available on the chip. The left-hand side of the figure shows the actual wiring on the chip and the right-hand side shows the symbolic diagram of the network configuration. The darkened tiles have been used for implementing the thresholds. Several other topologies like feedback networks and networks with localized receptive fields can be configured with this chip.

## The complete system

Figure 10 shows how the neural network chip fits into a complete system that is necessary for its use and testing. The 'Config-EPROM' stores the bit pattern corre-

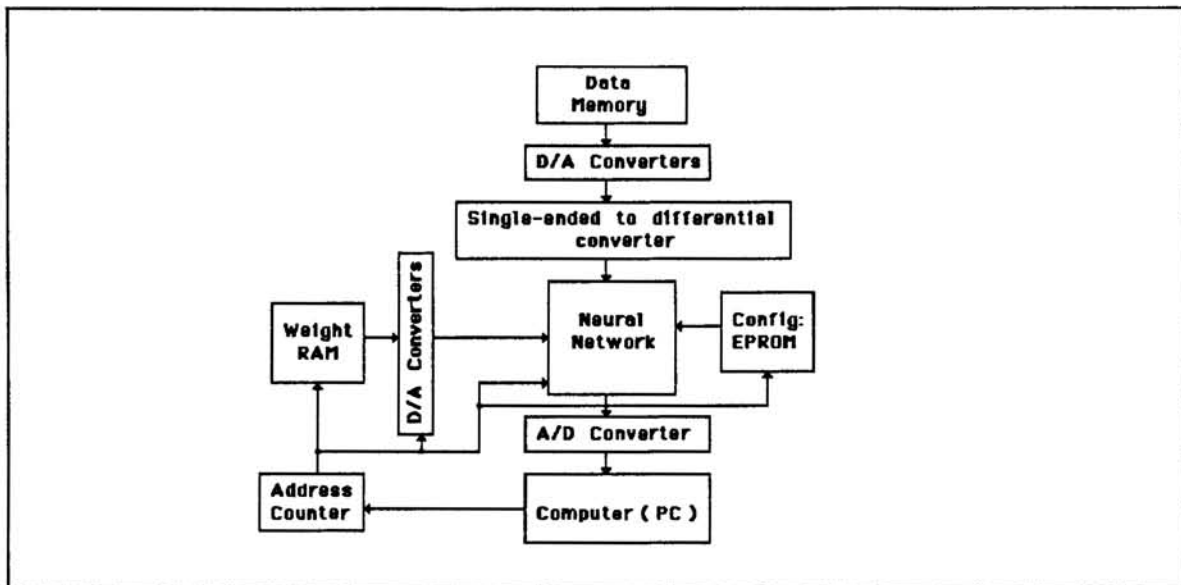

**Figure 10:** Block diagram of the system for reconfiguration, weight update/refresh and testing.

sponding to the desired topology. This bit pattern is down-loaded into the memory cells of the switch matrices before the start of computation. Input vectors are read out from the 'Data memory' and converted into analog form by D/A converters. The outputs of the D/A converters are further transformed into differential signals and then fed into the chip. The chip delivers differential outputs which are converted into digital form using an A/D converter and stored in a computer for further analysis.

The delay in processing one layer with N inputs driving another layer with an equal number of inputs is typically $1\mu$sec. Hence a 12-32-12 network should take about $6\mu$secs for one forward-pass operation. However external loads can slow down the computation considerably. This problem can be solved by increasing the bias currents or/and using pad buffers. Each block on the chip has been tested and has been found to function as expected. Tests of the complete chip in a variety of neural network configurations are being planned.

## Conclusions

We have designed a reconfigurable array of 1024 distributed-neuron synapses that can be configured into several different types of neural networks. The distributed-neuron concept that is integral to this chip offers advantages in terms of modularity and automatic gain normalization . The chip can be cascaded with several other chips of the same type to build larger systems.

## References

[1] Richard Lippmann. Pattern classification using neural networks. *IEEE Communications Magazine*, 27(11):47–64, November 1989.

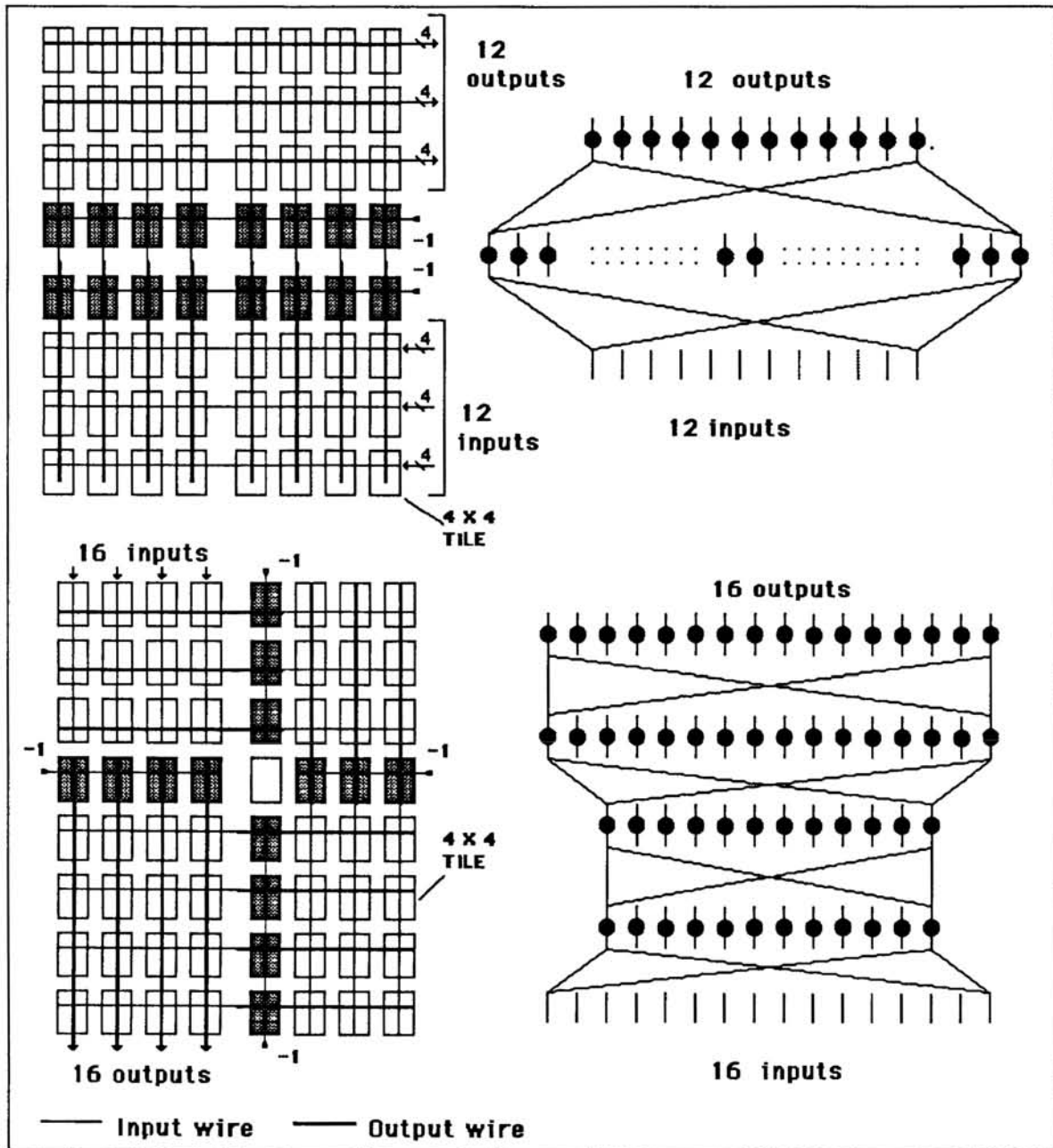

Figure 11: Reconfiguring the network to produce two different topologies

[2] Y. Tsividis and S. Satyanarayana. Analogue circuits for variable-synapse electronic neural networks. *Electronics Letters*, 23(24):1313–1314, November 1987.

[3] Y. Tsividis and D. Anastassiou. Switched-capacitor neural networks. *Electronics Letters*, 23(18):958–959, August 1987.

[4] Paul Mueller et al. *A Programmable Analog Neural Computer and Simulator*, volume 1 of *Advances in Neural Information Processing systems*, pages 712–719. Morgan Kaufmann Publishers, 1989.

[5] D. B. Schwartz, R. E. Howard, and W. E. Hubbard. *Adaptive Neural Networks Using MOS Charge Storage*, volume 1 of *Advances in Neural Information Processing systems*, pages 761–768. Morgan Kaufmann Publishers, 1989.

[6] J. R. Mann and S. Gilbert. *An Analog Self-Organizing Neural Network Chip*, volume 1 of *Advances in Neural Information Processing systems*, pages 739–747. Morgan Kaufmann Publishers, 1989.

[7] Mark Holler, Simon Tam, Hernan Castro, and Ronald Benson. An electrically trainable artificial neural network etann with 10240 'floating gate' synapses. In *IJCNN International Joint Conference on Neural Networks*, volume 2, pages 191–196. International Neural Network Society (INNS) and Institue of Electrical and Electronic Engineers (IEEE), 1989.

[8] S. Eberhardt, T. Duong, and A. Thakoor. Design of parallel hardware neural network systems from custom analog vlsi 'building block' chips. In *IJCNN International Joint Conference on Neural Networks*, volume 2, pages 191–196. International Neural Network Society (INNS) and Institue of Electrical and Electronic Engineers (IEEE), 1989.

[9] F. J. Kub, I. A. Mack, K. K. Moon, C. Yao, and J. Modola. Programmable analog synapses for microelectronic neural networks using a hybrid digital-analog approach. In *IEEE International Conference on Neural Networks, San Diego*, 1988.

[10] Y. Le Cun et al. Handwritten digit recognition: Application of neural network chips and automatic learning. *IEEE Communications Magazine*, 27(11):41–46, November 1989.

[11] S. Satyanarayana, Y. Tsividis, and H. P. Graf. Analogue neural networks with distributed neurons. *Electronics Letters*, 25(5):302–304, March 1989.
